# Improved Risk Tail Bounds
# for On-Line Algorithms *

**Nicolò Cesa-Bianchi**
DSI, Università di Milano
via Comelico 39
20135 Milano, Italy
cesa-bianchi@dsi.unimi.it

**Claudio Gentile**
DICOM, Università dell'Insubria
via Mazzini 5
21100 Varese, Italy
gentile@dsi.unimi.it

## Abstract

We prove the strongest known bound for the risk of hypotheses selected from the ensemble generated by running a learning algorithm incrementally on the training data. Our result is based on proof techniques that are remarkably different from the standard risk analysis based on uniform convergence arguments.

## 1 Introduction

In this paper, we analyze the risk of hypotheses selected from the ensemble obtained by running an arbitrary on-line learning algorithm on an i.i.d. sequence of training data. We describe a procedure that selects from the ensemble a hypothesis whose risk is, with high probability, at most

$$M_n + O\left(\frac{(\ln n)^2}{n} + \sqrt{\frac{M_n}{n}\ln n}\right),$$

where $M_n$ is the average cumulative loss incurred by the on-line algorithm on a training sequence of length $n$. Note that this bound exhibits the "fast" rate $(\ln n)^2/n$ whenever the cumulative loss $nM_n$ is $O(1)$.

This result is proven through a refinement of techniques that we used in [2] to prove the substantially weaker bound $M_n + O\big(\sqrt{(\ln n)/n}\big)$. As in the proof of the older result, we analyze the empirical process associated with a run of the on-line learner using exponential inequalities for martingales. However, this time we control the large deviations of the on-line process using Bernstein's maximal inequality rather than the Azuma-Hoeffding inequality. This provides a much tighter bound on the average risk of the ensemble. Finally, we relate the risk of a specific hypothesis within the ensemble to the average risk. As in [2], we select this hypothesis using a deterministic sequential testing procedure, but the use of Bernstein's inequality makes the analysis of this procedure far more complicated.

The study of the statistical risk of hypotheses generated by on-line algorithms, initiated by Littlestone [5], uses tools that are sharply different from those used for uniform convergence analysis, a popular approach based on the manipulation of suprema of empirical

processes (see, e.g., [3]). Unlike uniform convergence, which is tailored to empirical risk minimization, our bounds hold for *any* learning algorithm. Indeed, disregarding efficiency issues, any learner can be run incrementally on a data sequence to generate an ensemble of hypotheses.

The consequences of this line of research to kernel and margin-based algorithms have been presented in our previous work [2].

**Notation.** An *example* is a pair $(x, y)$, where $x \in \mathcal{X}$ (which we call *instance*) is a data element and $y \in \mathcal{Y}$ is the *label* associated with it. Instances $x$ are tuples of numerical and/or symbolic attributes. Labels $y$ belong to a finite set of symbols (the class elements) or to an interval of the real line, depending on whether the task is classification or regression. We allow a learning algorithm to output hypotheses of the form $h : \mathcal{X} \to \mathcal{D}$, where $\mathcal{D}$ is a decision space not necessarily equal to $\mathcal{Y}$. The goodness of hypothesis $h$ on example $(x, y)$ is measured by the quantity $\ell(h(x), y)$, where $\ell : \mathcal{D} \times \mathcal{Y} \to \mathbb{R}$ is a nonnegative and bounded *loss function*.

## 2 A bound on the average risk

An on-line algorithm A works in a sequence of trials. In each trial $t = 1, 2, \dots$ the algorithm takes in input a hypothesis $H_{t-1}$ and an example $Z_t = (X_t, Y_t)$, and returns a new hypothesis $H_t$ to be used in the next trial. We follow the standard assumptions in statistical learning: the sequence of examples $Z^n = \big((X_1, Y_1), \dots, (X_n, Y_n)\big)$ is drawn i.i.d. according to an unknown distribution over $\mathcal{X} \times \mathcal{Y}$. We also assume that the loss function $\ell$ satisfies $0 \le \ell \le 1$. The success of a hypothesis $h$ is measured by the *risk* of $h$, denoted by $\texttt{risk}(h)$. This is the expected loss of $h$ on an example $(X, Y)$ drawn from the underlying distribution, $\texttt{risk}(h) = \mathbb{E}\, \ell(h(X), Y)$. Define also $\texttt{risk}_{\texttt{emp}}(h)$ to be the empirical risk of $h$ on a sample $Z^n$,

$$\texttt{risk}_{\texttt{emp}}(h) = \frac{1}{n} \sum_{t=1}^{n} \ell(h(X_t), Y_t) \, .$$

Given a sample $Z^n$ and an on-line algorithm A, we use $H_0, H_1, \dots, H_{n-1}$ to denote the *ensemble of hypotheses generated by* A. Note that the ensemble is a function of the random training sample $Z^n$. Our bounds hinge on the sample statistic

$$M_n = M_n(Z^n) = \frac{1}{n} \sum_{t=1}^{n} \ell(H_{t-1}(X_t), Y_t)$$

which can be easily computed as the on-line algorithm is run on $Z^n$.

The following bound, a consequence of Bernstein's maximal inequality for martingales due to Freedman [4], is of primary importance for proving our results.

**Lemma 1** *Let $L_1, L_2, \dots$ be a sequence of random variables, $0 \le L_t \le 1$. Define the bounded martingale difference sequence $V_t = \mathbb{E}[L_t \mid L_1, \dots, L_{t-1}] - L_t$ and the associated martingale $S_n = V_1 + \dots + V_n$ with conditional variance $K_n = \sum_{t=1}^{n} \text{Var}[L_t \mid L_1, \dots, L_{t-1}]$. Then, for all $s, k \ge 0$,*

$$\mathbb{P}\left(S_n \ge s, K_n \le k\right) \le \exp\left(-\frac{s^2}{2k + 2s/3}\right) \, .$$

The next proposition, derived from Lemma 1, establishes a bound on the average risk of the ensemble of hypotheses.

**Proposition 2** *Let $H_0, \ldots, H_{n-1}$ be the ensemble of hypotheses generated by an arbitrary on-line algorithm A. Then, for any $0 < \delta \le 1$,*

$$\mathbb{P}\left(\frac{1}{n}\sum_{t=1}^{n}\mathtt{risk}(H_{t-1}) \ge M_n + \frac{36}{n}\ln\left(\frac{n\,M_n+3}{\delta}\right) + 2\sqrt{\frac{M_n}{n}\ln\left(\frac{n\,M_n+3}{\delta}\right)}\right) \le \delta\,.$$

The bound shown in Proposition 2 has the same rate as a bound recently proven by Zhang [6, Theorem 5]. However, rather than deriving the bound from Bernstein inequality as we do, Zhang uses an ad hoc argument.

*Proof.* Let

$$\mu_n = \frac{1}{n}\sum_{t=1}^{n}\mathtt{risk}(H_{t-1}) \quad \text{and} \quad V_{t-1} = \mathtt{risk}(H_{t-1}) - \ell(H_{t-1}(X_t), Y_t) \quad \text{for } t \ge 1.$$

Let $\kappa_t$ be the conditional variance $\mathrm{Var}\big(\ell(H_{t-1}(X_t), Y_t) \mid Z_1, \ldots, Z_{t-1}\big)$. Also, set for brevity $K_n = \sum_{t=1}^{n}\kappa_t$, $K'_n = \lfloor\sum_{t=1}^{n}\kappa_t\rfloor$, and introduce the function $A(x) = 2\ln\frac{(x+1)(x+3)}{\delta}$ for $x \ge 0$. We find upper and lower bounds on the probability

$$\mathbb{P}\left(\sum_{t=1}^{n}V_{t-1} \ge A(K_n) + \sqrt{A(K_n)\,K_n}\right). \tag{1}$$

The upper bound is determined through a simple stratification argument over Lemma 1. We can write

$$\mathbb{P}\left(\sum_{t=1}^{n}V_{t-1} \ge A(K_n) + \sqrt{A(K_n)\,K_n}\right)$$

$$\le \mathbb{P}\left(\sum_{t=1}^{n}V_{t-1} \ge A(K'_n) + \sqrt{A(K'_n)\,K'_n}\right)$$

$$\le \sum_{s=0}^{n}\mathbb{P}\left(\sum_{t=1}^{n}V_{t-1} \ge A(s) + \sqrt{A(s)\,s},\ K'_n = s\right)$$

$$\le \sum_{s=0}^{n}\mathbb{P}\left(\sum_{t=1}^{n}V_{t-1} \ge A(s) + \sqrt{A(s)\,s},\ K_n \le s+1\right)$$

$$\le \sum_{s=0}^{n}\exp\left(-\frac{(A(s) + \sqrt{A(s)\,s})^2}{\frac{2}{3}(A(s) + \sqrt{A(s)\,s}) + 2(s+1)}\right) \quad \text{(using Lemma 1).}$$

Since $\dfrac{(A(s)+\sqrt{A(s)\,s})^2}{\frac{2}{3}\big(A(s)+\sqrt{A(s)\,s}\big)+2(s+1)} \ge A(s)/2$ for all $s \ge 0$, we obtain

$$(1) \le \sum_{s=0}^{n}e^{-A(s)/2} = \sum_{s=0}^{n}\frac{\delta}{(s+1)(s+3)} < \delta. \tag{2}$$

As far as the lower bound on (1) is concerned, we note that our assumption $0 \le \ell \le 1$ implies $\kappa_t \le \mathtt{risk}(H_{t-1})$ for all $t$ which, in turn, gives $K_n \le n\mu_n$. Thus

$$(1) = \mathbb{P}\left(n\mu_n - nM_n \ge A(K_n) + \sqrt{A(K_n)\,K_n}\right)$$

$$\ge \mathbb{P}\left(n\mu_n - nM_n \ge A(n\mu_n) + \sqrt{A(n\mu_n)\,n\mu_n}\right)$$

$$= \mathbb{P}\left(2n\mu_n \ge 2nM_n + 3A(n\mu_n) + \sqrt{4n\,M_n\,A(n\mu_n) + 5A(n\mu_n)^2}\right)$$

$$= \mathbb{P}\left(x \ge B + \tfrac{3}{2}A(x) + \sqrt{B\,A(x) + \tfrac{5}{4}A^2(x)}\right),$$

where we set for brevity $x = n\mu_n$ and $B = n\,M_n$. We would like to solve the inequality

$$x \geq B + \tfrac{3}{2}A(x) + \sqrt{B\,A(x) + \tfrac{5}{4}A^2(x)} \qquad (3)$$

w.r.t. $x$. More precisely, we would like to find a suitable upper bound on the (unique) $x^*$ such that the above is satisfied as an equality.

A (tedious) derivative argument along with the upper bound $A(x) \leq 4\ln\left(\frac{x+3}{\delta}\right)$ show that

$$x' = B + 2\sqrt{B\ln\left(\frac{B+3}{\delta}\right)} + 36\ln\left(\frac{B+3}{\delta}\right)$$

makes the left-hand side of (3) larger than its right-hand side. Thus $x'$ is an upper bound on $x^*$, and we conclude that

$$(1) \geq \mathbb{P}\left(x \geq B + 2\sqrt{B\ln\left(\frac{B+3}{\delta}\right)} + 36\ln\left(\frac{B+3}{\delta}\right)\right)$$

which, recalling the definitions of $x$ and $B$, and combining with (2), proves the bound. □

## 3 Selecting a good hypothesis from the ensemble

If the decision space $\mathcal{D}$ of A is a convex set and the loss function $\ell$ is convex in its first argument, then via Jensen's inequality we can directly apply the bound of Proposition 2 to the risk of the *average hypothesis* $\overline{H} = \frac{1}{n}\sum_{t=1}^{n} H_{t-1}$ . This yields

$$\mathbb{P}\left(\texttt{risk}(\overline{H}) \geq M_n + \frac{36}{n}\ln\left(\frac{n\,M_n+3}{\delta}\right) + 2\sqrt{\frac{M_n}{n}\ln\left(\frac{n\,M_n+3}{\delta}\right)}\right) \leq \delta. \qquad (4)$$

Observe that this is a $O(1/n)$ bound whenever the cumulative loss $n\,M_n$ is $O(1)$.

If the convexity hypotheses do not hold (as in the case of classification problems), then the bound in (4) applies to a hypothesis randomly drawn from the ensemble (this was investigated in [1] though with different goals).

In this section we show how to deterministically pick from the ensemble a hypothesis whose risk is close to the average ensemble risk.

To see how this could be done, let us first introduce the functions

$$\mathcal{E}_\delta(r,t) = \frac{8B}{3(n-t)} + \sqrt{\frac{2Br}{n-t}} \qquad \text{and} \qquad c_\delta(r,t) = \mathcal{E}_\delta\left(r + \sqrt{\frac{2Br}{n-t}}, t\right),$$

with $B = \ln\frac{n(n+2)}{\delta}$.

Let $\texttt{risk}_{\texttt{emp}}(H_t, t+1) + \mathcal{E}_\delta\left(\texttt{risk}_{\texttt{emp}}(H_t, t+1), t\right)$ be the *penalized empirical risk* of hypothesis $H_t$, where

$$\texttt{risk}_{\texttt{emp}}(H_t, t+1) = \frac{1}{n-t}\sum_{i=t+1}^{n}\ell(H_t(X_i), Y_i)$$

is the empirical risk of $H_t$ on the remaining sample $Z_{t+1}, \ldots, Z_n$. We now analyze the performance of the learning algorithm that returns the hypothesis $\widehat{H}$ minimizing the penalized risk estimate over all hypotheses in the ensemble, i.e., [1]

$$\widehat{H} = \operatorname*{argmin}_{0 \leq t < n}\left(\texttt{risk}_{\texttt{emp}}(H_t, t+1) + \mathcal{E}_\delta\left(\texttt{risk}_{\texttt{emp}}(H_t, t+1), t\right)\right). \qquad (5)$$

**Lemma 3** *Let $H_0, \ldots, H_{n-1}$ be the ensemble of hypotheses generated by an arbitrary on-line algorithm $\mathbf{A}$ working with a loss $\ell$ satisfying $0 \leq \ell \leq 1$. Then, for any $0 < \delta \leq 1$, the hypothesis $\widehat{H}$ satisfies*

$$\mathbb{P}\left(\mathtt{risk}(\widehat{H}) > \min_{0 \leq t < n}\left(\mathtt{risk}(H_t) + 2\,c_\delta(\mathtt{risk}(H_t), t)\right)\right) \leq \delta \ .$$

*Proof.* We introduce the following short-hand notation

$$R_t \;=\; \mathtt{risk_{emp}}(H_t, t+1), \qquad \widehat{T} = \operatorname*{argmin}_{0 \leq t < n}\left(R_t + \mathcal{E}_\delta(R_t, t)\right)$$

$$T^* \;=\; \operatorname*{argmin}_{0 \leq t < n}\left(\mathtt{risk}(H_t) + 2c_\delta(\mathtt{risk}(H_t), t)\right) \ .$$

Also, let $H^* = H_{T^*}$ and $R^* = \mathtt{risk_{emp}}(H_{T^*}, T^*+1) = R_{T^*}$. Note that $\widehat{H}$ defined in (5) coincides with $H_{\widehat{T}}$. Finally, let

$$Q(r, t) = \frac{\sqrt{2B(2B + 9r(n-t))} - 2B}{3(n-t)} \ .$$

With this notation we can write

$$\mathbb{P}\left(\mathtt{risk}(\widehat{H}) > \mathtt{risk}(H^*) + 2c_\delta(\mathtt{risk}(H^*), T^*)\right)$$

$$\leq \quad \mathbb{P}\left(\mathtt{risk}(\widehat{H}) > \mathtt{risk}(H^*) + 2c_\delta\big(R^* - Q(R^*, T^*), T^*\big)\right)$$

$$+ \quad \mathbb{P}\left(\mathtt{risk}(H^*) < R^* - Q(R^*, T^*)\right)$$

$$\leq \quad \mathbb{P}\left(\mathtt{risk}(\widehat{H}) > \mathtt{risk}(H^*) + 2c_\delta\big(R^* - Q(R^*, T^*), T^*\big)\right)$$

$$+ \quad \sum_{t=0}^{n-1} \mathbb{P}\left(\mathtt{risk}(H_t) < R_t - Q(R_t, t)\right) \ .$$

Applying the standard Bernstein's inequality (see, e.g., [3, Ch. 8]) to the random variables $R_t$ with $|R_t| \leq 1$ and expected value $\mathtt{risk}(H_t)$, and upper bounding the variance of $R_t$ with $\mathtt{risk}(H_t)$, yields

$$\mathbb{P}\left(\mathtt{risk}(H_t) < R_t - \frac{B + \sqrt{B(B + 18(n-t)\mathtt{risk}(H_t))}}{3(n-t)}\right) \leq e^{-B} \ .$$

With a little algebra, it is easy to show that

$$\mathtt{risk}(H_t) < R_t - \frac{B + \sqrt{B(B + 18(n-t)\mathtt{risk}(H_t))}}{3(n-t)}$$

is equivalent to $\mathtt{risk}(H_t) < R_t - Q(R_t, t)$. Hence, we get

$$\mathbb{P}\left(\mathtt{risk}(\widehat{H}) > \mathtt{risk}(H^*) + 2c_\delta(\mathtt{risk}(H^*), T^*)\right)$$

$$\leq \quad \mathbb{P}\left(\mathtt{risk}(\widehat{H}) > \mathtt{risk}(H^*) + 2c_\delta\big(R^* - Q(R^*, T^*), T^*\big)\right) + n\,e^{-B}$$

$$\leq \quad \mathbb{P}\left(\mathtt{risk}(\widehat{H}) > \mathtt{risk}(H^*) + 2\mathcal{E}_\delta(R^*, T^*)\right) + n\,e^{-B}$$

where in the last step we used

$$Q(r,t) \leq \sqrt{\frac{2Br}{n-t}} \qquad \text{and} \qquad c_\delta\left(r - \sqrt{\frac{2Br}{n-t}}, t\right) = \mathcal{E}_\delta(r,t).$$

Set for brevity $\mathcal{E} = \mathcal{E}_\delta(R^*, T^*)$. We have

$$\mathbb{P}\left(\text{risk}(\widehat{H}) > \text{risk}(H^*) + 2\mathcal{E}\right)$$

$$= \mathbb{P}\left(\text{risk}(\widehat{H}) > \text{risk}(H^*) + 2\mathcal{E}, \ R_{\widehat{T}} + \mathcal{E}_\delta(R_{\widehat{T}}, \widehat{T}) \leq R^* + \mathcal{E}\right)$$

(since $R_{\widehat{T}} + \mathcal{E}_\delta(R_{\widehat{T}}, \widehat{T}) \leq R^* + \mathcal{E}$ holds with certainty)

$$\leq \sum_{t=0}^{n-1} \mathbb{P}\left(R_t + \mathcal{E}_\delta(R_t, t) \leq R^* + \mathcal{E}, \ \text{risk}(H_t) > \text{risk}(H^*) + 2\mathcal{E}\right). \quad (6)$$

Now, if $R_t + \mathcal{E}_\delta(R_t, t) \leq R^* + \mathcal{E}$ holds, then at least one of the following three conditions
$R_t \leq \text{risk}(H_t) - \mathcal{E}_\delta(R_t, t), \quad R^* > \text{risk}(H^*) + \mathcal{E}, \quad \text{risk}(H_t) - \text{risk}(H^*) < 2\mathcal{E}$
must hold. Hence, for any fixed $t$ we can write

$$\mathbb{P}\left(R_t + \mathcal{E}_\delta(R_t, t) \leq R^* + \mathcal{E}, \ \text{risk}(H_t) > \text{risk}(H^*) + 2\mathcal{E}\right)$$

$$\leq \mathbb{P}\left(R_t \leq \text{risk}(H_t) - \mathcal{E}_\delta(R_t, t), \ \text{risk}(H_t) > \text{risk}(H^*) + 2\mathcal{E}\right)$$

$$+ \mathbb{P}\left(R^* > \text{risk}(H^*) + \mathcal{E}, \ \text{risk}(H_t) > \text{risk}(H^*) + 2\mathcal{E}\right)$$

$$+ \mathbb{P}\left(\text{risk}(H_t) - \text{risk}(H^*) < 2\mathcal{E}, \ \text{risk}(H_t) > \text{risk}(H^*) + 2\mathcal{E}\right)$$

$$\leq \mathbb{P}\left(R_t \leq \text{risk}(H_t) - \mathcal{E}_\delta(R_t, t)\right) + \mathbb{P}\left(R^* > \text{risk}(H^*) + \mathcal{E}\right). \quad (7)$$

Plugging (7) into (6) we have

$$\mathbb{P}\left(\text{risk}(\widehat{H}) > \text{risk}(H^*) + 2\mathcal{E}\right)$$

$$\leq \sum_{t=0}^{n-1} \mathbb{P}\left(R_t \leq \text{risk}(H_t) - \mathcal{E}_\delta(R_t, t)\right) + n\,\mathbb{P}\left(R^* > \text{risk}(H^*) + \mathcal{E}\right)$$

$$\leq n\,e^{-B} + n \sum_{t=0}^{n-1} \mathbb{P}\left(R_t \geq \text{risk}(H_t) + \mathcal{E}_\delta(R_t, t)\right) \leq n\,e^{-B} + n^2\,e^{-B},$$

where in the last two inequalities we applied again Bernstein's inequality to the random variables $R_t$ with mean $\text{risk}(H_t)$. Putting together we obtain

$$\mathbb{P}\left(\text{risk}(\widehat{H}) > \text{risk}(H^*) + 2c_\delta(\text{risk}(H^*), T^*)\right) \leq (2n + n^2)e^{-B}$$

which, recalling that $B = \ln\frac{n(n+2)}{\delta}$, implies the thesis. $\qquad\square$

Fix $n \geq 1$ and $\delta \in (0,1)$. For each $t = 0, \ldots, n-1$, introduce the function

$$f_t(x) = x + \frac{11C}{3}\frac{\ln(n-t) + 1}{n-t} + 2\sqrt{\frac{2Cx}{n-t}}, \qquad x \geq 0,$$

where $C = \ln\frac{2n(n+2)}{\delta}$. Note that each $f_t$ is monotonically increasing. We are now ready to state and prove the main result of this paper.

**Theorem 4** *Fix any loss function $\ell$ satisfying $0 \leq \ell \leq 1$. Let $H_0, \ldots, H_{n-1}$ be the ensemble of hypotheses generated by an arbitrary on-line algorithm $\mathtt{A}$ and let $\widehat{H}$ be the hypothesis minimizing the penalized empirical risk expression obtained by replacing $c_\delta$ with $c_{\delta/2}$ in (5). Then, for any $0 < \delta \leq 1$, $\widehat{H}$ satisfies*

$$\mathbb{P}\left( \mathtt{risk}(\widehat{H}) \geq \min_{0 \leq t < n} f_t \left( M_{t,n} + \frac{36}{n-t} \ln \frac{2n(n+3)}{\delta} + 2 \sqrt{\frac{M_{t,n} \ln \frac{2n(n+3)}{\delta}}{n-t}} \right) \right) \leq \delta,$$

*where* $M_{t,n} = \frac{1}{n-t} \sum_{i=t+1}^{n} \ell(H_{i-1}(X_i), Y_i)$. *In particular, upper bounding the minimum over $t$ with $t = 0$ yields*

$$\mathbb{P}\left( \mathtt{risk}(\widehat{H}) \geq f_0 \left( M_n + \frac{36}{n} \ln \frac{2n(n+3)}{\delta} + 2 \sqrt{\frac{M_n \ln \frac{2n(n+3)}{\delta}}{n}} \right) \right) \leq \delta. \quad (8)$$

*For $n \to \infty$, bound (8) shows that $\mathtt{risk}(\widehat{H})$ is bounded with high probability by*

$$M_n + O\left( \frac{\ln^2 n}{n} + \sqrt{\frac{M_n \ln n}{n}} \right).$$

*If the empirical cumulative loss $n\, M_n$ is small (say, $M_n \leq c/n$, where $c$ is constant with $n$), then our penalized empirical risk minimizer $\widehat{H}$ achieves a $O\big((\ln^2 n)/n\big)$ risk bound. Also, recall that, in this case, under convexity assumptions the average hypothesis $\overline{H}$ achieves the sharper bound $O(1/n)$.*

*Proof.* Let $\mu_{t,n} = \frac{1}{n-t} \sum_{i=t}^{n-1} \mathtt{risk}(H_i)$. Applying Lemma 3 with $c_{\delta/2}$ we obtain

$$\mathbb{P}\left( \mathtt{risk}(\widehat{H}) > \min_{0 \leq t < n} \big( \mathtt{risk}(H_t) + c_{\delta/2}(\mathtt{risk}(H_t), t) \big) \right) \leq \frac{\delta}{2}. \quad (9)$$

We then observe that

$$\min_{0 \leq t < n} \Big( \mathtt{risk}(H_t) + c_{\delta/2}(\mathtt{risk}(H_t), t) \Big)$$

$$= \min_{0 \leq t < n} \min_{t \leq i < n} \Big( \mathtt{risk}(H_i) + c_{\delta/2}(\mathtt{risk}(H_i), i) \Big)$$

$$\leq \min_{0 \leq t < n} \frac{1}{n-t} \sum_{i=t}^{n-1} \Big( \mathtt{risk}(H_i) + c_{\delta/2}(\mathtt{risk}(H_i), i) \Big)$$

$$\leq \min_{0 \leq t < n} \left( \mu_{t,n} + \frac{1}{n-t} \sum_{i=t}^{n-1} \frac{8}{3} \frac{C}{n-i} + \frac{1}{n-t} \sum_{i=t}^{n-1} \left( \sqrt{\frac{2C\, \mathtt{risk}(H_i)}{n-i}} + \frac{C}{n-i} \right) \right)$$

(using the inequality $\sqrt{x+y} \leq \sqrt{x} + \frac{y}{2\sqrt{x}}$ )

$$= \min_{0 \leq t < n} \left( \mu_{t,n} + \frac{1}{n-t} \sum_{i=t}^{n-1} \frac{11}{3} \frac{C}{n-i} + \frac{1}{n-t} \sum_{i=t}^{n-1} \sqrt{\frac{2C\, \mathtt{risk}(H_i)}{n-i}} \right)$$

$$\leq \min_{0 \leq t < n} \left( \mu_{t,n} + \frac{11C}{3} \frac{\ln(n-t)+1}{n-t} + 2\sqrt{\frac{2C\mu_{t,n}}{n-t}} \right)$$

(using $\sum_{i=1}^{k} 1/i \leq 1 + \ln k$ and the concavity of the square root)

$$= \min_{0 \leq t < n} f_t(\mu_{t,n}).$$

Now, it is clear that Proposition 2 can be immediately generalized to imply the following set of inequalities, one for each $t = 0, \ldots, n-1$,

$$\mathbb{P}\left(\mu_{t,n} \geq M_{t,n} + \frac{36\,A}{n-t} + 2\sqrt{\frac{M_{t,n}\,A}{n-t}}\right) \leq \frac{\delta}{2n} \tag{10}$$

where $A = \ln\frac{2n(n+3)}{\delta}$. Introduce the random variables $K_0, \ldots, K_{n-1}$ to be defined later. We can write

$$\mathbb{P}\left(\min_{0 \leq t < n}\left(\texttt{risk}(H_t) + c_{\delta/2}(\texttt{risk}(H_t), t)\right) \geq \min_{0 \leq t < n} K_t\right)$$

$$\leq \mathbb{P}\left(\min_{0 \leq t < n} f_t(\mu_{t,n}) \geq \min_{0 \leq t < n} K_t\right) \leq \sum_{t=0}^{n-1} \mathbb{P}\left(f_t(\mu_{t,n}) \geq K_t\right).$$

Now, for each $t = 0, \ldots, n-1$, define $K_t = f_t\left(M_{t,n} + \frac{36\,A}{n-t} + 2\sqrt{\frac{M_{t,n}\,A}{n-t}}\right)$. Then (10) and the monotonicity of $f_0, \ldots, f_{n-1}$ allow us to obtain

$$\mathbb{P}\left(\min_{0 \leq t < n}\left(\texttt{risk}(H_t) + c_{\delta/2}(\texttt{risk}(H_t), t)\right) \geq \min_{0 \leq t < n} K_t\right)$$

$$\leq \sum_{t=0}^{n-1} \mathbb{P}\left(f_t(\mu_{t,n}) \geq f_t\left(M_{t,n} + \frac{36\,A}{n-t} + 2\sqrt{\frac{M_{t,n}\,A}{n-t}}\right)\right)$$

$$= \sum_{t=0}^{n-1} \mathbb{P}\left(\mu_{t,n} \geq M_{t,n} + \frac{36\,A}{n-t} + 2\sqrt{\frac{M_{t,n}\,A}{n-t}}\right) \leq \delta/2.$$

Combining with (9) concludes the proof. □

## 4 Conclusions and current research issues

We have shown tail risk bounds for specific hypotheses selected from the ensemble generated by the run of an arbitrary on-line algorithm. Proposition 2, our simplest bound, is proven via an easy application of Bernstein's maximal inequality for martingales, a quite basic result in probability theory. The analysis of Theorem 4 is also centered on the same martingale inequality. An open problem is to simplify this analysis, possibly obtaining a more readable bound. Also, the bound shown in Theorem 4 contains $\ln n$ terms. We do not know whether these logarithmic terms can be improved to $\ln(M_n n)$, similarly to Proposition 2. A further open problem is to prove lower bounds, even in the special case when $n M_n$ is bounded by a constant.

## Footnotes

*Part of the results contained in this paper have been presented in a talk given at the NIPS 2004 workshop on "(Ab)Use of Bounds".

[1]Note that, from an algorithmic point of view, this hypothesis is fairly easy to compute. In particular, if the underlying on-line algorithm is a standard kernel-based algorithm, $\widehat{H}$ can be calculated via a single sweep through the example sequence.

## References

[1] A. Blum, A. Kalai, and J. Langford. Beating the hold-out. In *Proc. 12th COLT*, 1999.

[2] N. Cesa-Bianchi, A. Conconi, and C. Gentile. On the generalization ability of on-line learning algorithms. *IEEE Trans. on Information Theory*, 50(9):2050–2057, 2004.

[3] L. Devroye, L. Győrfi, and G. Lugosi. *A Probabilistic Theory of Pattern Recognition*. Springer Verlag, 1996.

[4] D. A. Freedman. On tail probabilities for martingales. *The Annals of Probability*, 3:100–118, 1975.

[5] N. Littlestone. From on-line to batch learning. In *Proc. 2nd COLT*, 1989.

[6] T. Zhang. Data dependent concentration bounds for sequential prediction algorithms. In *Proc. 18th COLT*, 2005.
